# Combined Weak Classifiers

**Chuanyi Ji and Sheng Ma**
Department of Electrical, Computer and System Engineering
Rensselaer Polytechnic Institute, Troy, NY 12180
chuanyi@ecse.rpi.edu, shengm@ecse.rpi.edu

## Abstract

To obtain classification systems with both good generalization performance and efficiency in space and time, we propose a learning method based on combinations of weak classifiers, where weak classifiers are linear classifiers (perceptrons) which can do a little better than making random guesses. A randomized algorithm is proposed to find the weak classifiers. They are then combined through a majority vote. As demonstrated through systematic experiments, the method developed is able to obtain combinations of weak classifiers with good generalization performance and a fast training time on a variety of test problems and real applications.

## 1 Introduction

The problem we will investigate in this work is how to develop a classifier with both good generalization performance and efficiency in space and time in a supervised learning environment. The generalization performance is measured by the probability of classification error of a classifier. A classifier is said to be efficient if its size and the (average) time needed to develop such a classifier scale nicely (polynomially) with the dimension of the feature vectors, and other parameters in the training algorithm.

The method we propose to tackle this problem is based on combinations of weak classifiers[8][6], where the weak classifiers are the classifiers which can do a little better than random guessing. It has been shown by Schapire and Freund [8][4] that the computational power of weak classifiers is equivalent to that of a well-trained classifier, and an algorithm has been given to boost the performance of weak classifiers. What has not been investigated is the type of weak classifiers that can be used and how to find them. In practice, the ideas have been applied with success in hand-written character recognition to boost the performance of an already well-trained classifier. But the original idea on combining a large number of weak classifiers has not been used in solving real problems. An independent work

by Kleinberg[6] suggests that in addition to a good generalization performance, combinations of weak classifiers also provide advantages in computation time, since weak classifiers are computationally easier to obtain than well-trained classifiers. However, since the proposed method is based on an assumption which is difficult to realize, discrepancies have been found between the theory and the experimental results[7]. The recent work by Breiman[1][2] also suggests that combinations of classifiers can be computationally efficient, especially when used to learn large data sets.

The focus of this work is to investigate the following problems: (1) how to find weak classifiers, (2) what are the performance and efficiency of combinations of weak classifiers, and (3) what are the advantages of using combined weak classifiers compared with other pattern classification methods?

We will develop a randomized algorithm to obtain weak classifiers. We will then provide simulation results on both synthetic real problems to show capabilities and efficiency of combined weak classifiers. The extended version of this work with some of the theoretical analysis can be found in [5].

## 2  Weak Classifiers

In the present work, we choose linear classifiers (perceptrons) as weak classifiers. Let $\frac{1}{2} - \frac{1}{\nu}$ be the required generalization error of a classifier, where $\nu \geq 2$, is called the weakness factor which is used to characterize the strength of a classifier. The larger the $\nu$, the weaker the weak classifier. A set of weak classifiers are combined through a simple majority vote.

## 3  Algorithm

Our algorithm for combinations of weak classifiers consists of two steps: (1) generating individual weak classifiers through a simple randomized algorithm; and (2) combining a collection of weak classifiers through a simple majority vote.

Three parameters need to be chosen a priori for the algorithm: a weakness factor $\nu$, a number $\theta$ ($\frac{1}{2} \leq \theta < 1$) which will be used as a threshold to partition the training set, and the number of weak classifiers $2L + 1$ to be generated, where $L$ is a positive integer.

### 3.1  Partitioning the Training Set

The method we use to partition a training set is motivated by what given in [4]. Suppose a combined classifier consists of $K$ ($K \geq 1$) weak classifiers already. In order to generate a (new) weak classifier, the entire training set of $N$ training samples is partitioned into two subsets: a set of $M_1$ samples which contain all the misclassified samples and a small fraction of samples correctly-classified by the existing combined classifier; and the remaining $N - M_1$ training samples. The set of $M_1$ samples are called "cares", since they will be used to select a new weak classifier, while the rest of the samples are the "don't-cares".

The threshold $\theta$ is used to determine which samples should be assigned as cares. For instance, for the $n$-th training sample ($1 \leq n \leq N$), the performance index $a(n)$ is recorded, where $a(n)$ is the fraction of the weak classifiers in the existing combined classifier which classify the $n$-th sample correctly. If $a(n) < \theta$, this sample is assigned to the cares. Otherwise, it is a don't-care. This is done for all $N$ samples.

Through partitioning a training set in this way, a newly-generated weak classifier is forced to learn the samples which have not been learned by the existing weak classifiers. In the meantime, a properly-chosen $\theta$ can ensure that enough samples are used to obtain each weak classifier.

## 3.2   Random Sampling

To achieve a fast training time, we obtain a weak classifier by randomly sampling the classifier-space of all possible linear classifiers.

Assume that a feature vector $x \in R^d$ is distributed over a compact region $D$. The direction of a hyperplane characterized by a linear classifier with a weight vector, is first generated by randomly selecting the elements of the weight vector based on a uniform distribution over $(-1, 1)^d$. Then the threshold of the hyperplane is determined by randomly picking an $x \in D$, and letting the hyperplane pass through $x$. This will generate random hyperplanes which pass through the region $D$, and whose directions are randomly distributed in all directions. Such a randomly selected classifier will then be tested on all the cares. If it misclassifies a fraction of cares no more than $\frac{1}{2} - \frac{1}{\nu} - \epsilon$ ($\epsilon > 0$ and small), the classifier is kept and will be used in the combination. Otherwise, it is discarded. This process is repeated until a weak classifier is finally obtained.

A newly-generated weak classifier is then combined with the existing ones through a simple majority vote. The entire training set will then be tested on the combined classifier to result in a new set of cares, and don't-cares. The whole process will be repeated until the total number $2L + 1$ of weak classifiers are generated. The algorithm can be easily extended to multiple classes. Details can be found in [5].

## 4   Experimental Results

Extensive simulations have been carried out on both synthetic and real problems using our algorithm. One synthetic problem is chosen to test the efficiency of our method. Real applications from standard data bases are selected to compare the generalization performance of combinations of weak classifiers (CW) with that of other methods such as K-Nearest-Neighbor classifiers (K-NN)[1], artificial neural networks (ANN), combinations of neural networks (CNN), and stochastic discriminations (SD).

### 4.1   A Synthetic Problem: Two Overlapping Gaussians

To test the scaling properties of combinations of weak classifiers, a non-linearly separable problem is chosen from a standard database called ELENA [2]. The problem is a two-class classification problem, where the distributions of samples in both classes are multi-variate Gaussians with the same mean but different variances for each independent variable. There is a considerable amount of overlap between the samples in two classes, and the problem is non-linearly separable. The average generalization error and the standard deviations are given in Figure 1 for our algorithm based on 20 runs, and for other classifiers. The Bayes error is also given to show the theoretical limit. The results show that the performance of kNN degrades very quickly. The performance of ANN is better than that of kNN but still deviates more and more from the Bayes error as $d$ gets large. The combination of weak classifiers

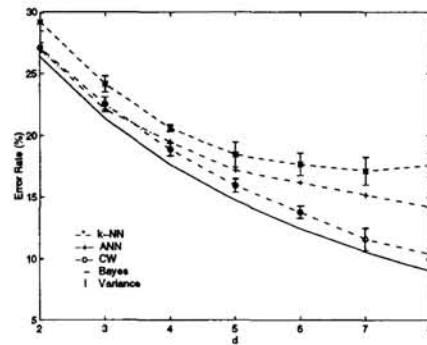

Figure 1: Performance versus the dimension of the feature vectors

| Algorithms | Card1 | Diabetes1 | Gene1 |
|---|---|---|---|
| | (%) Error /$\sigma$ | (%) Error /$\sigma$ | (%) Error /$\sigma$ |
| Combined Weak Classifiers | 11.3/ 0.85 | 22.70 / 0.70 | 11.80 / 0.52 |
| k Nearest Neighbor | 15.67 | 25.8 | 22.87 |
| Neural Networks | 13.64/ 0.85 | 23.52/ 0.72 | 13.47/ 0.44 |
| Combined Neural Networks | 13.02/0.33 | 22.79 /0.57 | 12.08 / 0.23 |

Table 1: Performance on Card1, Diabetes1 and Gene1. $\sigma$: standard deviation

continues to follow the trend of the Bayes error.

## 4.2 Proben1 Data Sets

Three data sets, Card1, Diabetes1 and Gene1 were selected to test our algorithm from Proben1 databases which contain data sets from real applications[3].

Card1 data set is for a problem on determining whether a credit-card application from a customer can be approved based on information given in 51-dimensional feature vectors. 345 out of 690 examples are used for training and the rest for testing. Diabetes1 data set is for determining whether diabetes is present based on 8-dimensional input patterns. 384 examples are used for training and the same number of samples for testing. Gene1 data set is for deciding whether a DNA sequence is from a donor, an acceptor or neither from 120 dimensional binary feature vectors. 1588 samples out of total of 3175 were used for training, and the rest for testing.

The average generalization error as well as the standard deviations are reported in Table 1. The results from combinations of weak classifiers are based on 25 runs. The results of neural networks and combinations of well-trained neural networks are from the database. As demonstrated by the results, combinations of weak classifiers have been able to achieve the generalization performance comparable to or better than that of combinations of well-trained neural networks.

## 4.3 Hand-written Digit Recognition

Hand-written digit recognition is chosen to test our algorithm, since one of the previously developed method on combinations of weak classifiers (stochastic discrimination[6]) was applied to this problem. For the purpose of comparison, the

| Algorithms | (%) Error/$\sigma$ |
|---|---|
| Combined Weak Classifiers | 4.23 / 0.1 |
| k Nearest Neighbor | 4.84 |
| Neural Networks | 5.33 |
| Stochastic Discriminations | 3.92 |

Table 2: Performance on handwritten digit recognition.

| Parameters | Gaussians | Card1 | Diabetes1 | Gene1 | Digits |
|---|---|---|---|---|---|
| $1/2 + 1/\nu$ | 0.51 | 0.51 | 0.51 | 0.55 | 0.54 |
| $\theta$ | 0.51 | 0.51 | 0.54 | 0.54 | 0.53 |
| 2L+1 | 2000 | 1000 | 1000 | 4000 | 20000 |
| Average Tries | 2 | 3 | 7 | 4 | 2 |

Table 3: Parameters used in our experiments.

same set of data as used in [6](from the NIST data base) is utilized to train and to test our algorithm. The data set contains 10000 digits written by different people. Each digit is represented by 16 by 16 black and white pixels. The first 4997 digits are used to form a training set, and the rest are for testing. Performance of our algorithm, k-NN, neural networks, and stochastic discriminations are given in Table 2. The results for our methods are based on 5 runs, while the results for the other methods are from [6]. The results show that the performance of our algorithm is slightly worse (by 0.3%) than that of stochastic discriminations, which uses a different method for multi-class classification[6].

## 4.4 Effects of The Weakness Factor

Experiments are done to test the effects of $\nu$ on the problem of two 8-dimensional overlapping Gaussians. The performance and the average training time (CPU-time on Sun Spac-10) of combined weak classifiers based on 10 runs are given for different $\nu$'s in Figures 2 and 3, respectively. The results indicate as $\nu$ increases an individual weak classifier is obtained more quickly, but more weak classifiers are needed to achieve good performance. When a proper $\nu$ is chosen, a nice scaling property can be observed in training time.

A record of the parameters used in all the experiments on real applications are provided in Table 3. The average tries, which are the average number of times needed to sample the classifier space to obtain an acceptable weak classifier, are also given in the table to characterize the training time for these problems.

## 4.5 Training Time

To compare learning time with off-line BackPropagation (BP), feedforward two layer neural network with 10 sigmoidal hidden units are trained by gradient-descent to learn the problem on the two 8-dimensional overlapping Gaussians. 2500 training samples are used. The performance versus CPU time[4] are plotted for both our algorithm and BP in Figure 4. For our algorithm, 2000 weak classifiers are combined. For BP, 1000 epochs are used. The figure shows that our algorithm is much faster than the BP algorithm. Moreover, when several well-trained neural networks are combined to achieve a better performance, the cost on training time will be

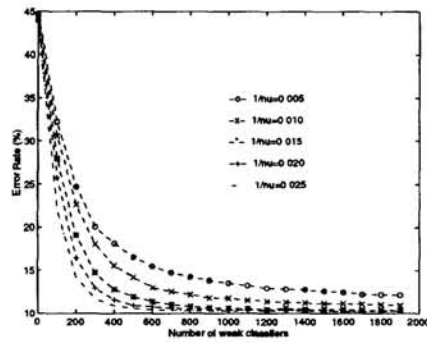

Figure 2: Performance versus the number of weak classifiers for different $\nu$. nu: $\nu$.

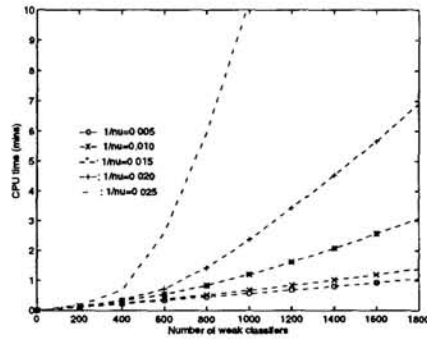

Figure 3: Training time versus the number of weak classifiers for different $\nu$.

even higher. Therefore, compared to combinations of well-trained neural networks, combining weak classifiers is computationally much cheaper.

## 5 Discussions

From the experimental results, we observe that the performance of the combined weak classifiers is comparable or even better than combinations of well-trained classifiers, and out-performs individual neural network classifiers and k-Nearest Neighbor classifiers. In the meantime whereas the $k$-nearest neighbor classifiers suffer from the curse of dimensionality, a nice scaling property in terms of the dimension of feature vectors has been observed for combined weak classifiers. Another

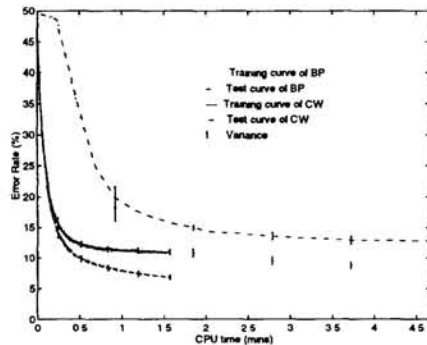

Figure 4: Performance versus CPU time

important observation obtained from the experiments is that the weakness factor directly impacts the size of a combined classifier and the training time. Therefore, the choice of the weakness factor is important to obtain efficient combined weak classifiers. It has been shown in our theoretical analysis on learning an underlying perceptron [5] that $\nu$ should be at least large as $O(dlnd)$ to obtain a polynomial training time, and the price paid to accomplish this is a space-complexity which is polynomial in $d$ as well. This cost can be observed from our experimental results for the need of a large number of weak classifiers.

## Acknowledgement

Specials thanks are due to Tin Kan Ho for providing NIST data, related references and helpful discussions. Support from the National Science Foundation (ECS-9312594 and (CAREER) IRI-9502518) is gratefully acknowledged.

## Footnotes

[1]The best result of different k is reported.

[2]/pub/neural-nets/ELENA/databases/Benchmarks.ps.Z on ftp.dice.ucl.ac.be

[3] Available by anonymous ftp from ftp.ira.uka.de, as /pub/papers/techreports/1994/1994-21.ps.z.

[4]Both algorithms are run on a Sun Sparc-10 sun workstation

## References

[1] L. Breiman, "Bias, Variance and Arcing Classifiers," *Technical Report, TR-460, Department of Statistics, University of California, Berkeley*, April, 1996.

[2] L. Breiman, "Pasting, Bites Together for Prediction in Large Data sets and On-Line," ftp.stat.berkeley.edu/users/breiman, 1996.

[3] H. Drucker, R. Schapire and P. Simard, "Improving Performance in Neural Networks Using a Boosting Algorithm," *Neural Information Processing Symposium*, 42-49, 1993.

[4] Y. Freund and R. Schapire, "A Decision-Theoretic Generalization of On-Line Learning and An Application to Boosting," http://www.research.att.com/orgs/ssr/people/yoav or schapire.

[5] C. Ji and S. Ma, "Combinations of Weak Classifiers," *IEEE Trans. Neural Networks, Special Issue on Neural Networks and Pattern Recognition*, vol. 8, 32-42, Jan., 1997.

[6] E.M. Kleinberg, "Stochastic Discrimination," *Annals of Mathematics and Artificial Intelligence*, vol.1, 207-239, 1990.

[7] E.M. Kleinberg and T. Ho, "Pattern Recognition by Stochastic Modeling," *Proceedings of the Third International Workshop on Frontiers in Handwriting Recognition*, 175-183, Buffalo, May 1993.

[8] R.E. Schapire, "The Strength of Weak Learnability," *Machine Learning*, vol. 5, 197-227, 1990.